# Circuit Optimization Predicts Dynamic Networks for Chemosensory Orientation in the Nematode *Caenorhabditis elegans*

**Nathan A. Dunn**     **John S. Conery**
Dept. of Computer Science
University of Oregon
Eugene, OR 97403
{ndunn,conery}@cs.uoregon.edu

**Shawn R. Lockery**
Institute of Neuroscience
University of Oregon
Eugene, OR 97403
shawn@lox.uoregon.edu *

## Abstract

The connectivity of the nervous system of the nematode *Caenorhabditis elegans* has been described completely, but the analysis of the neuronal basis of behavior in this system is just beginning. Here, we used an optimization algorithm to search for patterns of connectivity sufficient to compute the sensorimotor transformation underlying *C. elegans* chemotaxis, a simple form of spatial orientation behavior in which turning probability is modulated by the rate of change of chemical concentration. Optimization produced differentiator networks with inhibitory feedback among all neurons. Further analysis showed that feedback regulates the latency between sensory input and behavior. Common patterns of connectivity between the model and biological networks suggest new functions for previously identified connections in the *C. elegans* nervous system.

## 1 Introduction

The complete description of the morphology and synaptic connectivity of all 302 neurons in the nematode *Caenorhabditis elegans* [15] raised the prospect of the first comprehensive understanding of the neuronal basis of an animal's entire behavioral repertoire. The advent of new electrophysiological and functional imaging techniques for *C. elegans* neurons [7, 8] has made this project more realistic than before. Further progress would be accelerated, however, by prior knowledge of the sensorimotor transformations underlying the behaviors of *C. elegans*, together with knowledge of how these transformations could be implemented with *C. elegans*-like neuronal elements.

In previous work, we and others have identified the main features of the sensorimotor transformation underlying *C. elegans* chemotaxis [5, 11], one of two forms of spatial orientation identified in this species. Locomotion consists of periods of sinusoidal forward movement, called "runs," which are punctuated by bouts of turning [12] that have been termed "pirouettes" [11]. Pirouette probability is modulated by the rate of change of chemical concentration ($dC(t)/dt$). When $dC(t)/dt < 0$, pirouette probability is increased whereas when

$dC(t)/dt > 0$, pirouette probability is decreased. Thus, runs down the gradient are truncated and runs up the gradient are extended, resulting in net movement toward the gradient peak.

The process of identifying the neurons that compute this sensorimotor transformation is just beginning. The chemosensory neurons responsible for the input representation are known[1], as are the premotor interneurons for turning behavior[2]. Much less is known about the interneurons that link inputs to outputs. To gain insight into how this transformation might be computed at the interneuronal level, we used an unbiased parameter optimization algorithm to construct model neural networks capable of computing the transformation using *C. elegans*-like neurons. We found that networks with one or two interneurons were sufficient. A common but unexpected feature of all networks was inhibitory feedback among all neurons. We propose that the main function of this feedback is to regulate the latency between sensory input and behavior.

## 2   Assumptions

We used simulated annealing to search for patterns of connectivity sufficient for computing the chemotaxis sensorimotor transformation. The algorithm was constrained by three main assumptions:

1. Primary chemosensory neurons in *C. elegans* report attractant concentration at a single point in space.

2. Chemosensory interneurons converge on a network of locomotory command neurons to regulate turning probability.

3. The sensorimotor transformation in *C. elegans* is computed mainly at the network level, not at the cellular level.

Assumption (1) follows from the anatomy and distribution of chemosensory organs in *C. elegans*[1, 13, 14]. Assumption (2) follows from anatomical reconstructions of the *C. elegans* nervous system [15], together with the fact that laser ablation studies have identified four pairs of pre-motor interneurons that are necessary for turning in *C. elegans*[2]. Assumption (3) is heuristic.

## 3   Network

Neurons were modeled by the equation:

$$\tau_i \frac{dA_i(t)}{dt} = -A_i(t) + \sigma(I_i), \quad \text{with} \quad I_i = \sum_j (w_{ji} A_j(t)) + b_i \tag{1}$$

where $A_i$ is activation level of neuron $i$ in the network, $\sigma(I_i)$ is the logistic function $1/(1 + e^{-I_i})$, $w_{ji}$ is the synaptic strength from neuron $j$ to neuron $i$, and $b_i$ is static bias. The time constant $\tau_i$ determines how rapidly the activation approaches its steady-state value for constant $I_i$. Equation 1 embodies the additional assumption that, on the time scale of chemotaxis behavior, *C. elegans* neurons are effectively passive, isopotential nodes that release neurotransmitter in graded fashion. This assumption follows from preliminary electrophysiological recordings from neurons and muscles in *C. elegans* and *Ascaris*, another species of nematode[3, 4, 6].

The model of the chemosensory network had one input neuron, eight interneurons, and one output neuron (Figure 1). The input neuron ($i = 0$) was a lumped representation of all

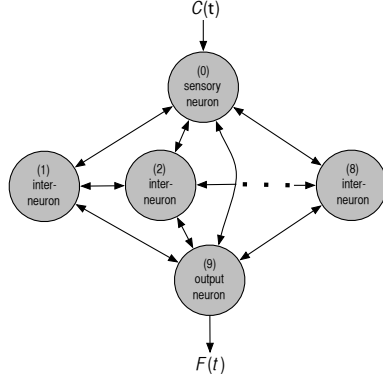

Figure 1: Model chemosensory network. Model neurons were passive, isopoential nodes. The network contained one sensory neuron, one output neuron, and eight interneurons. Input to the sensory neuron was the time course of chemoattractant concentration $C(t)$. The activation of the output neuron was mapped to turning probability by the function $F(t)$ given in Equation 2. The network was fully connected with self-connections (not shown).

the chemosensory neurons in the real animal. Sensory input to the network was $C(t)$, the time course of attractant concentration experienced by a real worm in an actual chemotaxis assay[11]. $C(t)$ was added to the net input of the sensory neuron ($i = 0$). The interneurons in the model ($1 \leq i \leq 8$) represented all the chemosensory interneurons in the real animal. The activity level of the output neuron ($i = 9$) determined the behavioral state of the model, i.e. turning probability[11], according to the piecewise function:

$$F(t) = \begin{cases} P_{high} & A_9(t) \leq T_1 \\ P_{rest} & T_1 < A_9(t) < T_2 \\ P_{low} & A_9(t) \geq T_2 \end{cases} \qquad (2)$$

where $T_1$ and $T_2$ are arbitrary thresholds and the three $P$ values represent the indicated levels of turning probability.

## 4 Optimization

The chemosensory network model was optimized to compute an idealized version of the true sensorimotor transformation linking $C(t)$ to turning probability[11]. To construct the idealized transformation, we mapped the instantaneous derivative of $C(t)$ to desired turning probability $G(t)$ as follows:

$$G(t) = \begin{cases} P_{high} & dC(t)/dt \leq -U \\ P_{rest} & -U < dC(t)/dt < +U \\ P_{low} & dC(t)/dt \geq +U \end{cases} \qquad (3)$$

where $U$ is a threshold derived from previous behavioral observations (Figure 7 in [11]). The goal of the optimization was to make the network's turning probability $F(t)$ equal to the desired turning probability $G(t)$ at all $t$. Optimization was carried out by annealing three parameter types: weights, time constants, and biases. Optimized networks were fully connected and self-connections were allowed.

The result of a typical optimization run is illustrated in Figure 2(a), which shows good agreement between network and desired turning probabilities. Results similar to Figure 2(a) were found for 369 networks out of 401 runs (92%). We noted that in most networks, many interneurons had a constant offset but showed little or no response to changes in sensory input. We found that we could eliminate these interneurons by a pruning procedure in which the tonic effect of the offset was absorbed into the bias term of postsynaptic neurons. Pruning had little or no effect on network performance (Figure 2(b)), suggesting

that the eliminated neurons were nonfunctional. By this procedure, 67% of the networks could be reduced to one interneuron and 27% could be reduced to two interneurons. A key question is whether the network generalizes to a $C(t)$ time course that it has not seen before. Generalization was tested by challenging pruned networks with the $C(t)$ time course from a second real chemotaxis assay. There was good agreement between network and desired turning probability, indicating an acceptable level of generalization (Figure 2(c)).

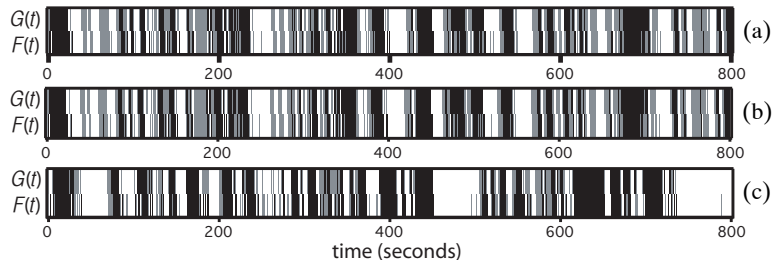

Figure 2: Network performance after optimization. In each panel, the upper trace represents $G(t)$, the desired turning probability in response to a particular $C(t)$ time course (not shown), whereas the lower trace represents $F(t)$, the resulting network turning probability. Shading signifies turning probability (black = $P_{high}$, grey = $P_{rest}$, white = $P_{low}$). (a) Performance of a typical network after optimization. (b) Performance of the same network after pruning. (c) Performance of the pruned network when stimulated by a different $C(t)$ time course. Network turning probability is delayed relative to desired turning probability because of the time required for sensory input to affect behavioral state.

## 5 Results

Here we focus on the largest class of networks, those with a single interneuron (Figure 3(a)). All single-interneuron networks had three common features (Figure 3(b)). First, the direct pathway from sensory neuron to output neuron was excitatory, whereas the indirect pathway via the interneuron was inhibitory. Such a circuit computes an approximate derivative of its input by subtracting a delayed version of the input from its present value[9]. Second, all neurons had significant inhibitory self-connections. We noted that inhibitory self-connections were strongest on the input and output neurons, the two neurons comprising the direct pathway representing current sensory input. We hypothesized that the function of inhibitory self-connections was to decrease response latency in the direct pathway. Such a decrease would be a means of compensating for the fact that $G(t)$ was an instantaneous function of $C(t)$, whereas the neuronal time constant $\tau_i$ tends to introduce a delay between $C(t)$ and the network's output. Third, the net effect of all disynaptic recurrent connections was also inhibitory. By analogy to inhibitory self-connections, we hypothesized that the function of the recurrent pathways was also to regulate response latency.

To test the hypothetical functions of the self-connections and recurrent connections, we introduced an explicit time delay ($\Delta t$) between $dC(t)/dt$ and the desired turning probability $G(t)$ such that:

$$G'(t) = G(t - \Delta t) \tag{4}$$

$G'(t)$ was then substituted for $G(t)$ during optimization. We then repeated the optimization procedure with a range of $\Delta t$ values and looked for systematic effects on connectivity.

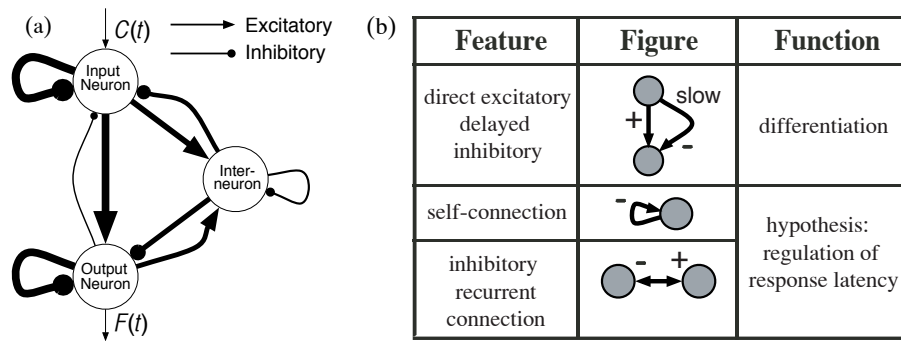

Figure 3: Connectivity and common features of single-interneuron networks. (a) Average sign and strength of connections. Line thickness is proportional to connection strength. In other single-interneuron networks, the sign of the connections to and from the interneuron were reversed (not shown). (b) The three common features of single-interneuron networks.

**Effects on self-connections.** We found that the magnitude of self-connections on the input and output neurons was inversely related to $\Delta t$ (Figure 4(a)). This result suggests that the function of these self-connections is to regulate response latency, as hypothesized. We noted that the interneuron self-connection remains comparatively small regardless of $\Delta t$. This result is consistent with the function of the disynaptic pathway, which is to present a delayed version of the input to the output neuron.

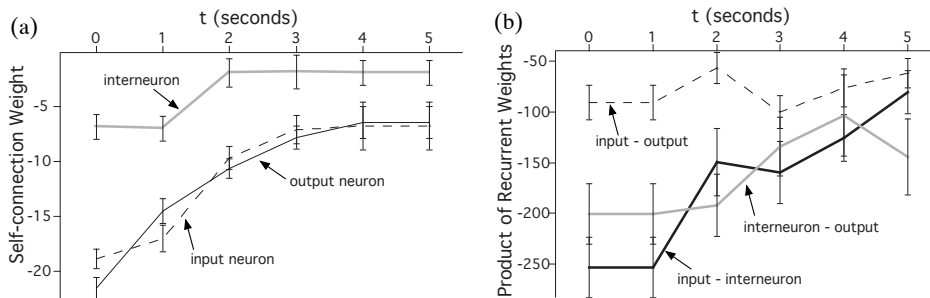

Figure 4: The effect on connectivity of introducing time delays between input and output during optimization. (a) The effect on self-connections. (b) The effect on recurrent connections. Recurrent connection strength was quantified by taking the product of the weights along each of the three recurrent loops in Figure 3(a).

**Effects on recurrent connections.** We quantified the strength of the recurrent connections by taking the product of the two weights along each of the three recurrent loops in the network. We found that the strengths of the two recurrent loops that included the interneuron was inversely related to $\Delta t$ (Figure 4(b)). This result suggests that the function of these loops is to regulate response latency and supports the hypothetical function of the recur-

rent connections. Interestingly, however, the strength of the recurrent loop between input and output neurons was not affected by changes in $\Delta t$. Comparing the overall patterns of changes in weights produced by changes in $\Delta t$ showed that the optimization algorithm utilized self-connections to adjust delays along the direct pathway and recurrent connections to adjust delays along the indirect pathway. The reason for this pattern is presently unclear.

## 6   Analysis

To provide a theoretical explanation for the effects of time delays on the magnitude of self-connections, we analyzed the step response of Equation 1 for a reduced system containing a single linear neuron with a self-connection:

$$\tau_i \frac{dA_i(t)}{dt} = w_{ii} A_i(t) - A_i(t) + h(t) \tag{5}$$

where $h(t)$ represents a generic external input (sensory or synaptic). Solving Equation 5 for $h(t)$ equal to a step of amplitude $M$ at $t = 0$ with $A(0) = 0$ gives:

$$A_i(t) = \left( \frac{M}{1 - w_{ii}} \right) \left[ 1 - exp \left[ - \left( \frac{1 - w_{ii}}{\tau_i} t \right) \right] \right] \tag{6}$$

From Equation 6, when $w_{ii} = 0$ (no self-connection) the neuron relaxes at the rate $1/\tau_i$, whereas when $w_{ii} < 0$ (inhibitory self-connection) the neuron relaxes at the higher rate of $(1 + |w_{ii}|)/\tau_i$. Thus, response latency drops as the strength of the inhibitory self connection increases and, conversely, response latency rises as connection strength decreases. This result explains the effect on self-connection strength of introducing a delay between between $dC(t)/dt$ and turning probability (Figure 4(a)).

We made a similar analysis of the effects of time delays on the recurrent connections. Here, however, we studied a reduced system of two linear neurons with reciprocal synapses and an external input to one of the neurons.

$$\tau_i \frac{dA_i(t)}{dt} = w_{ji} A_j(t) - A_i(t) + h(t) \qquad \text{and} \qquad \tau_j \frac{dA_j(t)}{dt} = w_{ij} A_i(t) - A_j(t) \tag{7}$$

We solved this system for the case where the external input $h(t) = M \sin(\Omega t)$. The solution has the form:

$$A_i(t) = D_i \sin(\Omega t - \phi_i) \qquad \text{and} \qquad A_j(t) = D_j \sin(\Omega t - \phi_j) \tag{8}$$

$$\text{with} \qquad \phi_i = \phi_j = \arctan \left[ \frac{2 \Omega \tau}{1 - w_{ij} w_{ji} - \Omega^2 \tau^2} \right] \tag{9}$$

Equation (9) gives the phase delay between the sinusoidal external input and the sinusoidal response of the two neuron system. In Figure 5, the relationship between phase delay and the strength of the recurrent connections is plotted with the connection strength on the ordinate as in Figure 4(b). The graph shows an inverse relationship between connection strength and phase delay that approximates the inverse relationship between connection strength and time delay shown in Figure 4(b). The correspondence between the trends in Figure 4(b) and 5 explain the effects on recurrent connection strength of introducing a delay between between $dC(t)/dt$ and turning probability.

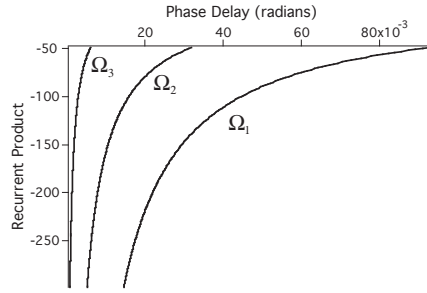

Figure 5: The relationship between phase delay and recurrent connection strength. Equation 9 is plotted for three different driving frequencies, (Hz $\times 10^{-3}$): $\Omega_1 = 50$, $\Omega_2 = 18.75$, and $\Omega_3 = 3.75$. These frequencies span the frequencies observed in a Fourier analysis of the $C(t)$ time course used during optimization. There is an inverse relationship between connection strength and phase delay. Axis have been reversed for comparison with Figure 4(b).

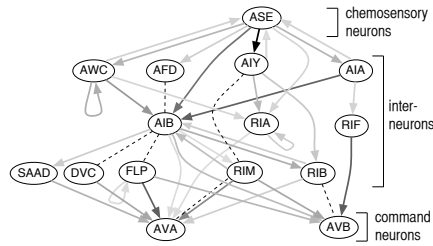

Figure 6: The network of chemosensory interneurons in the real animal. Shown are the interneurons interposed between the chemosensory neuron ASE and the two locomotory command neurons AVA and AVB. The diagram is restricted to interneuron pathways with less than three synapses. Arrows are chemical synapses. Dashed lines are gap junctions. Connectivity is inferred from the anatomical reconstructions of reference [15].

## 7  Discussion

We used simulated annealing to search for networks capable of computing an idealized version of the chemotaxis sensorimotor transformation in *C. elegans*. We found that one class of such networks is the three neuron differentiator with inhibitory feedback. The appearance of differentiator networks was not surprising [9] because the networks were optimized to report, in essence, the sign of $dC(t)/dt$ (Equation 3). The prevalence of inhibitory feedback, however, was unexpected. Inhibitory feedback was found at two levels: self-connections and recurrent connections. Combining an empirical and theoretical approach, we have argued that inhibitory feedback at both levels functions to regulate the response latency of the system's output relative to its input. Such regulation could be functionally significant in the *C. elegans* nervous system, where neurons may have an unusually high input resistance due to their small size. High input resistance could lead to long relaxation times because the membrane time constant is proportional to input resistance. The types of inhibitory feedback identified here could also be used to mitigate this effect.

There are intriguing parallels between our three-neuron network models and the biological network. Figure 6 shows the network of interneurons interposed between the chemosensory neuron class ASE, the main chemosensory neurons for salt chemotaxis, and the locomotory command neurons classes AVB and AVA. The interneurons in Figure 6 are candidates for computing the sensorimotor transformation for chemotaxis *C. elegans*. Three parallels are prominent. First, there are two candidate differentiator circuits, as noted previously[16]. These circuits are formed by the neuronal triplets ASE-AIA-AIB and ASE-AWC-AIB. Second, there are self-connections on three neuron classes in the circuit, including AWC, one of the differentiator neurons. These self-connections represent anatomically identified connections between left and right members of the respective classes. It remains to be seen, however, whether these connections are inhibitory in the biological network. Self-connections could also be implemented at the cellular level by voltage dependent currents. A voltage-dependent potassium current, for example, can be functionally equivalent to an

inhibitory self-connection. Electrophysiological recordings from ASE and other neurons in *C. elegans* confirm the presence of such currents[6, 10]. Thus, it is conceivable that many neurons in the biological network have the cellular equivalent of self-connections. Third, there are reciprocal connections between ASE and three of its four postsynaptic targets. These connections could provide recurrent inhibition if they have the appropriate signs.

Common patterns of connectivity between the model and biological networks suggest new functionality for identified connections in the *C. elegans* nervous system. It should be possible to test these functions through physiological recordings and neuronal ablations.

## Acknowledgements

We are grateful Don Pate for his technical assistance. Supported by NSF IBN-0080068.

## Footnotes

*To whom correspondence should be addressed.

## References

[1] C. I. Bargmann and H. R. Horvitz. Chemosensory neurons with overlapping functions direct chemotaxis to multiple chemicals in *C. elegans*. *Neuron*, 7:729–742, 1991.

[2] M. Chalfie, J.E. Sulston, J.G. White, E. Southgate, J.N. Thomson, and S. Brenner. The neural circuit for touch sensitivity in *C. elegans*. *J. of Neurosci.*, 5:956–964, 1985.

[3] R. E. Davis and A. O. Stretton. Passive membrane properties of motorneurons and their role in long-distance signaling in the nematode *Ascaris*. *J. of Neurosci.*, 9:403–414, 1989.

[4] R. E. Davis and A. O. W. Stretton. Signaling properties of *Ascaris* motorneurons: graded active response, graded synaptic transmission, and tonic transmitter release. *J. of Neurosci.*, 9:415–425, 1989.

[5] D.B. Dusenbery. Responses of the nematode *C. elegans* to controlled chemical stimulation. *J. of Comparative Physiology*, 136:127–331, 1980.

[6] M.B. Goodman, D.H. Hall, L. Avery, and S.R. Lockery. Active currents regulate sensitivity and dynamic range in *C. elegans* neurons. *Neuron*, 20:763–772, 1998.

[7] R. Kerr, V. Lev-Ram, G. Baird, P. Vincent, R. Y. Tsien, and W. R. Schafer. Optical imaging of calcium transients in neurons and pharyngeal muscle of *C. elegans*. *Neuron*, 26(3):583–94, 2000.

[8] S. R. Lockery and M. B. Goodman. Tight-seal whole-cell patch clamping of *C. elegans* neurons. *Methods Enzymol*, 293:201–17, 1998.

[9] E.E. Munro, L.E. Shupe, and E.E Fetz. Integration and differentiation in dynamical recurrent neural networks. *Neural Computation*, 6:405–419, 1994.

[10] W.T. Nickell, R.Y. Pun, C.I. Bargmann, and S.J. Kleene. Single ionic channels of two *C. elegans* chemosensory neurons in native membrane. *J. of Membrane Biology*, 189(1):55–66, 2002.

[11] J. T. Pierce-Shimomura, T. M. Morse, and S. R. Lockery. The fundamental role of pirouettes in *C. elegans* chemotaxis. *J. of Neurosci.*, 19(21):9557–69, 1999.

[12] T.A. Rutherford and N.A. Croll. Wave forms of *C. elegans* in a chemical attractant and repellent and in thermal gradients. *J. of Nematology*, 11:232–240, 1979.

[13] S. Ward. Chemotaxis by the nematode *C. elegans*: identification of attractants and analysis of the response by use of mutants. *Proc of the Natl Acad Sci USA*, 70:817–821, 1973.

[14] S. Ward, N. Thomson, J. G. White, and S. Brenner. Electron microscopical reconstruction of the anterior sensory anatomy of the nematode *C. elegans*. *J. of Comparative Neurology*, 160:313–338, 1975.

[15] J. G White, E. Southgate, J. N. Thomson, and S. Brenner. The structure of the nervous system of the nematode *C. elegans*. *Phil Trans of the R Soc Lond [Biol]*, 314:1–340, 1986.

[16] J.G. White. Neuronal connectivity in *C. elegans*. *Trends in Neuroscience*, 8:277–283, 1985.
